# Factorized Latent Spaces with Structured Sparsity

**Yangqing Jia[1], Mathieu Salzmann[1,2], and Trevor Darrell[1]**
[1]UC Berkeley EECS and ICSI   [2]TTI-Chicago
{jiayq,trevor}@eecs.berkeley.edu, salzmann@ttic.edu

## Abstract

Recent approaches to multi-view learning have shown that factorizing the information into parts that are shared across all views and parts that are private to each view could effectively account for the dependencies and independencies between the different input modalities. Unfortunately, these approaches involve minimizing non-convex objective functions. In this paper, we propose an approach to learning such factorized representations inspired by sparse coding techniques. In particular, we show that structured sparsity allows us to address the multi-view learning problem by alternately solving two convex optimization problems. Furthermore, the resulting factorized latent spaces generalize over existing approaches in that they allow having latent dimensions shared between any subset of the views instead of between all the views only. We show that our approach outperforms state-of-the-art methods on the task of human pose estimation.

## 1   Introduction

Many computer vision problems inherently involve data that is represented by multiple modalities such as different types of image features, or images and surrounding text. Exploiting these multiple sources of information has proven beneficial for many computer vision tasks. Given these multiple views, an important problem therefore is that of learning a latent representation of the data that best leverages the information contained in each input view.

Several approaches to addressing this problem have been proposed in the recent years. Multiple kernel learning [2, 24] methods have proven successful under the assumption that the views are independent. In contrast, techniques that learn a latent space shared across the views (Fig. 1(a)), such as Canonical Correlation Analysis (CCA) [12, 3], the shared Kernel Information Embedding model (sKIE) [23], and the shared Gaussian Process Latent Variable Model (shared GPLVM) [21, 6, 15], have shown particularly effective to model the dependencies between the modalities. However, they do not account for the independent parts of the views, and therefore either totally fail to represent them, or mix them with the information shared by all views.

To generalize over the above-mentioned approaches, methods have been proposed to explicitly account for the dependencies and independencies of the different input modalities. To this end, these methods factorize the latent space into a shared part common to all views and a private part for each modality (Fig. 1(b)). This has been shown for linear mappings [1, 11], as well as for non-linear ones [7, 14, 20]. In particular, [20] proposed to encourage the shared-private factorization to be non-redundant while simultaneously discovering the dimensionality of the latent space. The resulting FOLS models were shown to yield more accurate results in the context of human pose estimation. This, however, came at the price of solving a complicated, non-convex optimization problem. FOLS also lacks an efficient inference method, and extension from two views to multiple views is not straightforward since the number of shared/latent spaces that need to be explicitly modeled grows exponentially with the number of views.

In this paper, we propose a novel approach to finding a latent space in which the information is correctly factorized into shared and private parts, while avoiding the computational burden of previous techniques [14, 20]. Furthermore, our formulation has the advantage over existing shared-private factorizations of allowing shared information between any subset of the views, instead of only be-

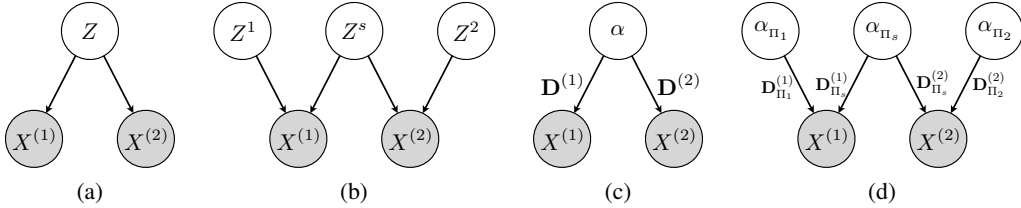

<center>(a)             (b)             (c)             (d)</center>

Figure 1: Graphical models for the two-view case of (a) shared latent space models [23, 21, 6, 15], (b) shared-private factorizations [7, 14, 20], (c) the global view of our model, where the shared-private factorization is automatically learned instead of explicitly separated, and (d) an equivalent shared-private spaces interpretation of our model. Due to structured sparsity, rows $\Pi_s$ of $\boldsymbol{\alpha}$ are shared across the views, whereas rows $\Pi_1$ and $\Pi_2$ are private to view 1 and 2, respectively.

tween all views. In particular, we represent each view as a linear combination of view-dependent dictionary entries. While the dictionaries are specific to each view, the weights of these dictionaries act as latent variables and are the same for all the views. Thus, as shown in Fig. 1(c), the data is embedded in a latent space that generates all the views. By exploiting the idea of structured sparsity [26, 18, 4, 17, 9], we encourage each view to only use a subset of the latent variables, and at the same time encourage the whole latent space to be low-dimensional. As a consequence, and as depicted in Fig. 1(d), the latent space is factorized into shared parts which represent information common to multiple views, and private parts which model the remaining information of the individual views. Training the model can be done by alternately solving two convex optimization problems, and inference by solving a convex problem.

We demonstrate the effectiveness of our approach on the problem of human pose estimation where the existence of shared and private spaces has been shown [7]. We show that our approach correctly factorizes the latent space and outperforms state-of-the-art techniques.

## 2   Learning a Latent Space with Structured Sparsity

In this section, we first formulate the problem of learning a latent space for multi-view modeling. We then briefly review the concepts of sparse coding and structured sparsity, and finally introduce our approach within this framework.

### 2.1   Problem Statement and Notations

Let $\mathcal{X} = \{\mathbf{X}^{(1)}, \mathbf{X}^{(2)}, \cdots, \mathbf{X}^{(V)}\}$ be a set of $N$ observations obtained from $V$ views, where $\mathbf{X}^{(v)} \in \Re^{P_v \times N}$ contains the feature vectors for the $v^{\text{th}}$ view. We aim to find an embedding $\boldsymbol{\alpha} \in \Re^{N_d \times N}$ of the data into an $N_d$-dimensional latent space and a set of dictionaries $\mathcal{D} = \{\mathbf{D}^{(1)}, \mathbf{D}^{(2)}, \cdots, \mathbf{D}^{(V)}\}$, with $\mathbf{D}^{(v)} \in \Re^{P_v \times N_d}$ the dictionary entries for view $v$, such that $\mathbf{X}^{(v)}$ is generated by $\mathbf{D}^{(v)}\boldsymbol{\alpha}$, as depicted in Fig. 1(c). More specifically, we seek the latent embedding $\boldsymbol{\alpha}$ and the dictionaries that best reconstruct the data in the least square sense by solving the optimization problem

$$\min_{\mathcal{D}, \boldsymbol{\alpha}} \sum_{v=1}^{V} \|\mathbf{X}^{(v)} - \mathbf{D}^{(v)}\boldsymbol{\alpha}\|_{\text{Fro}}^2 \ . \tag{1}$$

Furthermore, as explained in Section 1, we aim to find a latent space that naturally separates the information shared among several views from the information private to each view. Our approach to addressing this problem is inspired by structured sparsity, which we briefly review below.

Throughout this paper, given a matrix $\mathbf{A}$, we will use the term $\mathbf{A}_i$ to denote its $i^{\text{th}}$ column vector, $\mathbf{A}_{i,\cdot}$ to denote its $i^{\text{th}}$ row vector, and $\mathbf{A}_{\cdot,\Omega}$ ($\mathbf{A}_{\Omega,\cdot}$) to denote the submatrix formed by taking a subset of its columns (rows), where the set $\Omega$ contains the indices of the chosen columns (rows).

### 2.2   Sparse Coding and Structured Sparsity

In the single-view case, sparse coding techniques [16, 25, 13] have been proposed to represent the observed data (e.g., image features) as a linear combination of dictionary entries, while encouraging each observation vector to only employ a subset of all the available dictionary entries. More formally, let $\mathbf{X} \in \Re^{P \times N}$ be the matrix of training examples. Sparse coding aims to find a set of dictionary entries $\mathbf{D} \in \Re^{P \times N_d}$ and the corresponding linear combination weights $\boldsymbol{\alpha} \in \Re^{N_d \times N}$ by solving the optimization problem

$$\min_{\mathbf{D},\boldsymbol{\alpha}} \ \frac{1}{N}||\mathbf{X} - \mathbf{D}\boldsymbol{\alpha}||^2_{\text{Fro}} + \lambda\phi(\boldsymbol{\alpha}) \tag{2}$$
$$\text{s.t.} \ ||\mathbf{D}_i|| \leq 1 \ , \ 1 \leq i \leq N_d \ ,$$

where $\phi$ is a regularizer that encourages sparsity of its input, and $\lambda$ is the weight that sets the relative influence of both terms. In practice, when $\phi$ is a convex function, problem (2) is convex in $\mathbf{D}$ for a fixed $\boldsymbol{\alpha}$ and vice-versa. Typically, the $L_1$ norm is used to encourage sparsity, which yields

$$\phi(\boldsymbol{\alpha}) = \sum_{j=1}^{N}||\boldsymbol{\alpha}_j||_1 = \sum_{j=1}^{N}\sum_{i=1}^{N_d}|\boldsymbol{\alpha}_{i,j}| \ . \tag{3}$$

While sparse coding has proven effective in many domains, it fails to account for any structure in the observed data. For instance, in classification tasks, one would expect the observations belonging to the same class to depend on the same subset of dictionary entries. This problem has been addressed by structured sparse coding techniques [26, 4, 9], which encode the structure of the problem in the regularizer. Typically, these methods rely on the notion of groups among the training examples to encourage members of the same group to rely on the same dictionary entries. This can simply be done by re-writing problem (2) as

$$\min_{\mathbf{D},\boldsymbol{\alpha}} \ \frac{1}{N}||\mathbf{X} - \mathbf{D}\boldsymbol{\alpha}||^2_{\text{Fro}} + \lambda\sum_{g=1}^{N_g}\psi(\boldsymbol{\alpha}_{\cdot,\Omega_g}) \tag{4}$$
$$\text{s.t.} \ ||\mathbf{D}_i|| \leq 1 \ , \ 1 \leq i \leq N_d \ ,$$

where $N_g$ is the total number of groups, $\Omega_g$ represents the indices of the examples that belong to group $g$, and $\boldsymbol{\alpha}_{\cdot,\Omega_g}$ is the matrix containing the weights associated to these examples. To keep the problem convex in $\boldsymbol{\alpha}$, $\psi$ is usually taken either as the $L_{1,2}$ norm, or as the $L_{1,\infty}$ norm, which yield

$$\psi(\boldsymbol{\alpha}_{\cdot,\Omega_g}) = \sum_{i=1}^{N_d}||\boldsymbol{\alpha}_{i,\Omega_g}||_2 \ , \ \text{or} \ \psi(\boldsymbol{\alpha}_{\cdot,\Omega_g}) = \sum_{i=1}^{N_d}||\boldsymbol{\alpha}_{i,\Omega_g}||_\infty = \sum_{i=1}^{N_d}\max_{k\in\Omega_g}|\boldsymbol{\alpha}_{i,k}| \ . \tag{5}$$

In general, structured sparsity can lead to more meaningful latent embeddings than sparse coding. For example, [4] showed that the dictionary learned by grouping local image descriptors into images or classes achieved better accuracy than sparse coding for small dictionary sizes.

## 2.3 Multi-view Learning with Structured Sparsity

While the previous framework has proven successful for many tasks, it has only been applied to the single-view case. Here, we propose an approach to multi-view learning inspired by structured sparse coding techniques. To correctly account for the dependencies and independencies of the views, we cast the problem as that of finding a factorization of the latent space into subspaces that are shared across several views and subspaces that are private to the individual views. In essence, this can be seen as having each view exploiting only a subset of the dimensions of the global latent space, as depicted by Fig. 1(d). Note that this definition is in fact more general than the usual definition of shared-private factorizations [7, 14, 20], since it allows latent dimensions to be shared across any subset of the views rather than across all views only.

More formally, to find a shared-private factorization of the latent embedding $\boldsymbol{\alpha}$ that represents the multiple input modalities, we adopt the idea of structured sparsity and aim to find a set of dictionaries $\mathcal{D} = \{\mathbf{D}^{(1)}, \mathbf{D}^{(2)}, \cdots, \mathbf{D}^{(V)}\}$, each of which uses only a subspace of the latent space. This can be achieved by re-formulating problem (1) as

$$\min_{\mathcal{D},\boldsymbol{\alpha}} \ \frac{1}{N}\sum_{v=1}^{V}||\mathbf{X}^{(v)} - \mathbf{D}^{(v)}\boldsymbol{\alpha}||^2_{\text{Fro}} + \lambda\sum_{v=1}^{V}\psi((\mathbf{D}^{(v)})^T) \tag{6}$$
$$\text{s.t.} \ ||\boldsymbol{\alpha}_{\cdot,i}|| \leq 1 \ , \ 1 \leq i \leq N_d \ .$$

where the regularizer $\psi((\mathbf{D}^{(v)})^T)$ can be defined using the $L_{1,2}$ or $L_{1,\infty}$ norm. In practice, we chose the $L_{1,\infty}$ norm regularizer which has proven more effective than the $L_{1,2}$ [18, 17]. Note that, here, we enforce structured sparsity on the dictionary entries instead of on the weights $\boldsymbol{\alpha}$. Furthermore, note that this sparsity encourages the columns of the individual $\mathbf{D}^{(v)}$ to be zeroed-out instead of the rows in the usual formulation. The intuition behind this is that we expect each view $\mathbf{X}^{(v)}$ to only depend on a subset of the latent dimensions. Since $\mathbf{X}^{(v)}$ is generated by $\mathbf{D}^{(v)}\boldsymbol{\alpha}$, having zero-valued columns of $\mathbf{D}^{(v)}$ removes the influence of the corresponding latent dimensions on the reconstruction.

While the formulation in Eq. 6 encourages each view to only use a limited number of latent dimensions, it doesn't guarantee that parts of the latent space will be shared across the views. With a sufficiently large number $N_d$ of dictionary entries, the same information can be represented in several parts of the dictionary. This issue is directly related to the standard problem of finding the correct dictionary size. A simple approach would be to manually choose the dimension of the latent space, but this introduces an additional hyperparameter to tune. Instead, we propose to address this issue by trying to find the smallest size of dictionary that still allows us to reconstruct the data well. In spirit, the motivation is similar to [8, 20] that use a relaxation of rank constraints to discover the dimensionality of the latent space. Here, we further exploit structured sparsity and re-write problem (6) as

$$\min_{\mathcal{D},\boldsymbol{\alpha}} \frac{1}{N} \sum_{v=1}^{V} \|\mathbf{X}^{(v)} - \mathbf{D}^{(v)}\boldsymbol{\alpha}\|_{\text{Fro}}^2 + \lambda \sum_{v=1}^{V} \psi((\mathbf{D}^{(v)})^T) + \gamma\psi(\boldsymbol{\alpha}) \ , \tag{7}$$

where we replaced the constraints on $\boldsymbol{\alpha}$ by an $L_{1,\infty}$ norm regularizer that encourages rows of $\boldsymbol{\alpha}$ to be zeroed-out. This lets us automatically discover the dimensonality of the latent space $\boldsymbol{\alpha}$. Furthermore, if there is shared information between several views, this regularizer will favor representing it in a single latent dimension, instead of having redundant parts of the latent space.

The optimization problem (7) is convex in $\mathcal{D}$ for a fixed $\boldsymbol{\alpha}$ and vice versa. Thus, in practice, we alternate between optimizing $\mathcal{D}$ with a fixed $\boldsymbol{\alpha}$ and the opposite. Furthermore, to speed up the process, after each iteration, we remove the latent dimensions whose norm is less than a pre-defined threshold. Note that efficient optimization techniques for the $L_{1,\infty}$ norm have been proposed in the literature [17], enabling efficient optimization algorithms for the problem.

### 2.4 Inference

At inference, given a new observation $\{\mathbf{x}_*^{(1)}, \cdots, \mathbf{x}_*^{(V)}\}$, the corresponding latent embedding $\alpha_*$ can be obtained by solving the convex problem

$$\min_{\alpha_*} \sum_{v=1}^{V} \|\mathbf{x}_*^{(v)} - \mathbf{D}^{(v)}\alpha_*\|_2^2 + \gamma\|\alpha_*\|_1 \ , \tag{8}$$

where the regularizer lets us deal with noise in the observations.

Another advantage of our model is that it easily allows us to address the case where only a subset of the views are observed at test time. This scenario arises, for example, in human pose estimation, where view $\mathbf{X}^{(1)}$ corresponds to image features and view $\mathbf{X}^{(2)}$ contains the 3D poses. At inference, the goal is to estimate the pose $\mathbf{x}_*^{(2)}$ given new image features $\mathbf{x}_*^{(1)}$. To this end, we seek to estimate the latent variables $\alpha_*$, as well as the unknown views from the available views. This is equivalent to first solving the convex problem

$$\min_{\alpha_*} \sum_{v \in \mathcal{V}_a} \|\mathbf{x}_*^{(v)} - \mathbf{D}^{(v)}\alpha_*\|_2^2 + \gamma\|\alpha_*\|_1 \ , \tag{9}$$

where $\mathcal{V}_a$ is the set of indices of available views. The remaining unobserved views $\mathbf{x}_*^{(v)}$ , $v \notin \mathcal{V}_a$ are then estimated as $\mathbf{x}_*^{(v)} = \mathbf{D}^{(v)}\alpha_*$ .

## 3 Related Work

While our method is closely related to the shared-private factorization algorithms which we discussed in Section 1, it was inspired by the existing sparse coding literature and therefore is also

| Method | $\psi(\mathbf{D})$ | $\phi(\boldsymbol{\alpha})$ | $\mathcal{C}_{\mathbf{D}}$ or $\mathcal{C}_{\boldsymbol{\alpha}}$ |
|---|---|---|---|
| PCA | none | none | $\{\mathbf{D}\vert\mathbf{D}^T\mathbf{D}=\mathbf{I}\}$ |
| SC (e.g. [25]) | none | $\Vert\boldsymbol{\alpha}^T\Vert_{1,1}$ | $\{\mathbf{D}\vert\Vert\mathbf{D}_i\Vert_2 \leq 1\ \forall i \leq N_d\}$ |
| Group SC [4] | $\Vert\mathbf{D}^T\Vert_{1,2}$ | $\sum_{\Omega_g}\Vert\boldsymbol{\alpha}_{\cdot,\Omega_g}\Vert_{1,2}$ | none |
| SSPCA [9] | $\sum_{\Omega_g}\Vert\mathbf{D}_{\cdot,\Omega_g}\Vert_{\xi,2}$ † | none | $\{\boldsymbol{\alpha}\vert\Vert\boldsymbol{\alpha}_{i,\cdot}\Vert_2 \leq 1\ \forall i \leq N_d\}$ |
| Group Lasso [26] | none | $\sum_{\Omega_g}\Vert(\boldsymbol{\alpha}_{\Omega_g,\cdot})^T\Vert_{1,2}$ | $\{\mathbf{D}\vert\mathbf{D}^T\mathbf{D}=\mathbf{I}\}$ |
| Our Method | $\sum_{\Omega_g}\Vert(\mathbf{D}_{\Omega_g,\cdot})^T\Vert_{1,\infty}$ | $\Vert\boldsymbol{\alpha}\Vert_{1,\infty}$ | none |

† Here $\xi$ denotes the vector $l_\alpha/l_1$ quasi-norm. See [9] for details.

Table 1: Properties of the different algorithms that can be viewed as special cases of RMF.

related to it. In this section, we first show that many existing techniques can be considered as special cases of a general regularized matrix factorization (RMF) framework, and then discuss the relationships and differences between our method and the existing ones.

In general, the RMF problem can be defined as that of factorizing a $P \times N$ matrix $\mathbf{X}$ into the product of a $P \times M$ matrix $\mathbf{D}$ and an $M \times N$ matrix $\boldsymbol{\alpha}$ so that the residual error is minimized. Furthermore, RMF exploits structured or unstructured regularizers to constrain the forms of $\mathbf{D}$ and $\boldsymbol{\alpha}$. This can be expressed as the optimization problem

$$\min_{\mathbf{D},\boldsymbol{\alpha}}\ \frac{1}{N}\Vert\mathbf{X}-\mathbf{D}\boldsymbol{\alpha}\Vert_{\mathrm{Fro}}^2 + \lambda\psi(\mathbf{D}) + \gamma\phi(\boldsymbol{\alpha}) \qquad (10)$$
$$\text{s.t. } \mathbf{D}\in\mathcal{C}_{\mathbf{D}}\ ,\ \boldsymbol{\alpha}\in\mathcal{C}_{\boldsymbol{\alpha}}\ ,$$

where $\mathcal{C}_{\mathbf{D}}$ and $\mathcal{C}_{\boldsymbol{\alpha}}$ are the domains of the dictionary $\mathbf{D}$ and of latent embedding $\boldsymbol{\alpha}$, respectively. These domains allow to enforce additional constraints on those matrices. Several existing algorithms, such as PCA, sparse coding (SC), group SC, structured sparse PCA (SSPCA) and group Lasso, can be considered as special cases of this general framework. Table 1 lists the regularization terms and constraints used by these different algorithms.

Algorithms relying on structured sparsity exploit different types of matrix norm[1] to impose sparsity and different ways of grouping the rows or columns of $\mathbf{D}$ and $\boldsymbol{\alpha}$ using algorithm-specific knowledge. Group sparse coding [4] relies on supervised information such as class labels to define the groups, while in our case, we exploit the natural separation provided by the multiple views. As a result, while group sparse coding finds dictionary entries that encode class-related information, our method finds latent spaces factorized into subspaces shared among different views and subspaces private to the individual views.

Furthermore, while structured sparsity is typically enforced on $\boldsymbol{\alpha}$, our method employs it on the dictionary. This also is the case of [9] in their SSPCA algorithm. However, while in our approach the groups are taken as subsets of the rows of $\mathbf{D}$, their method follows the more usual approach of defining the groups as subsets of its columns. Their intuition for doing so was to encourage dictionary entries to represent the variability of parts of the observation space, such as the variability of the eyes in the context of face images.

Finally, it is worth noting that imposing structured sparsity regularization on both $\mathbf{D}$ and $\boldsymbol{\alpha}$ naturally yields a multi-view, multi-class latent space learning algorithm that can be deemed as a generalization of several algorithms summarized here.

## 4 Experimental Evaluation

In this section, we show the results of our approach on learning factorized latent spaces from multi-view inputs. We compare our results against those obtained with state-of-the-art techniques on the task of human pose estimation.

### 4.1 Toy Example

First, we evaluated our approach on the same toy case used by [20]. This shows our method's ability to correctly factorize a latent space into shared and private parts. This toy example consists of two

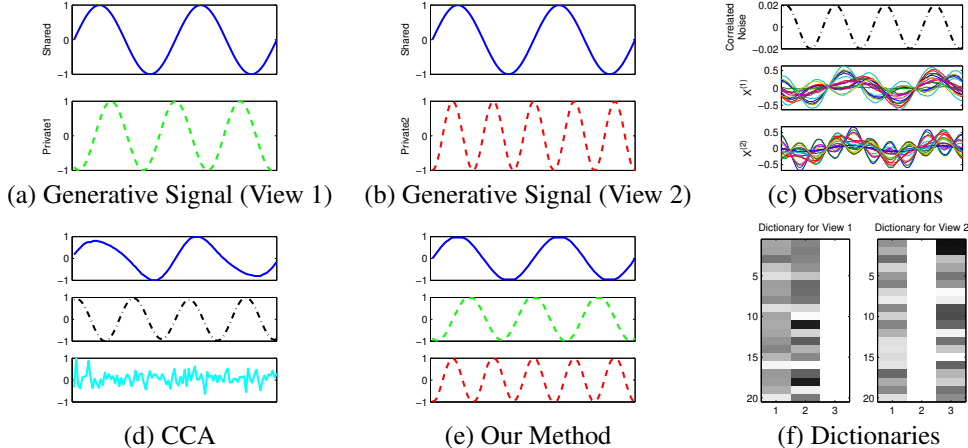

(a) Generative Signal (View 1)    (b) Generative Signal (View 2)    (c) Observations

(d) CCA            (e) Our Method          (f) Dictionaries

Figure 2: Latent spaces recovered on a toy example. (a,b) Generative signals for the two views. (c) Correlated noise and the two 20D input views. (d) First 3 dimensions recovered by CCA. (e) 3-dimensional latent space recovered with our method. Note that, as opposed to CCA, our approach correctly recovered the generative signals and discarded the noise. (f) Dictionaries learned by our algorithm for each view. Fully white columns correspond to zero-valued vectors; note that the dictionary for each view uses only the shared dimension and its own private dimension.

views generated from one shared signal and one private signal per view depicted by Fig. 2(a,b). In particular, we used sinusoidal signals at different frequencies such that

$$\boldsymbol{\alpha}^{(1)} = [\sin(2\pi\mathbf{t}); \cos(\pi^2\mathbf{t}))], \quad \boldsymbol{\alpha}^{(2)} = [\sin(2\pi\mathbf{t}); \cos(\sqrt{5}\pi\mathbf{t}))] \,, \tag{11}$$

where $\mathbf{t}$ was sampled from a uniform distribution in the interval $(-1, 1)$. This yields a 3-dimensional ground-truth latent space, with 1 shared dimension and 2 private dimensions. The observations $\mathbf{X}^{(v)}$ were generated by randomly projecting the $\boldsymbol{\alpha}^{(v)}$ into 20-dimensional spaces and adding Gaussian noise with variance 0.01. Finally, we added noise of the form $\mathbf{y}_{\text{noise}} = 0.02\sin(3.6\pi\mathbf{t})$ to both views to simulate highly correlated noise. The input views are depicted in Fig. 2(c)

To initialize our method, we first applied PCA separately on both views, as well as on the concatenation of the views, and in each case, kept the components representing 95% of the variance. We took $\boldsymbol{\alpha}$ as the concatenation of the corresponding weights. Note that the fact that this latent space is redundant is dealt with by our regularization on $\boldsymbol{\alpha}$. We then alternately optimized $\mathcal{D}$ and $\boldsymbol{\alpha}$, and let the algorithm determine the optimal latent dimensionality. Fig. 2(e,f) depicts the reconstructed latent spaces for both views, as well as the learned dictionaries, which clearly show the shared-private factorization. In Fig. 2(d), we show the results obtained with CCA. Note that our approach correctly discovered the original generative signals and discarded the noise, whereas CCA recovered the shared signal, but also the correlated noise and an additional noise. This confirms that our approach is well-suited to learn shared-private factorizations, and shows that CCA-based approaches [1, 11] tend to be sensitive to noise.

## 4.2 Human Pose Estimation

We then applied our method to the problem of human pose estimation, in which the task is to recover 3D poses from 2D image features. It has been shown that this problem is ambiguous, and that shared-private factorizations helped accounting for these ambiguities. Here, we used the HumanEva dataset [22] which consists of synchronized images and motion capture data describing the 3D locations of the 19 joints of a human skeleton. These two types of observations can be seen as two views of the same problem from which we can learn a latent space.

In our experiments, we compare our results with those of several regression methods that directly learn a mapping from image features to 3D poses. In particular, we used linear regression (Lin-Reg), Gaussian Process regression with a linear kernel (GP-lin) and with an RBF kernel (GP-rbf), and nearest-neighbor in the feature space (NN). We also compare our results with those obtained with the FOLS-GPLVM [20], which also proposes a shared-private factorization of the latent space. Note that we did not compare against other shared-private factorizations [7, 14], or purely shared

| Data | Lin-Reg | GP-lin | GP-rbf | NN | FOLS | Our Method |
|---|---|---|---|---|---|---|
| Jogging | 1.420 | 1.429 | 1.396 | 1.436 | 1.461 | 0.954 |
| Walking | 2.167 | 2.363 | 2.330 | 2.175 | 2.137 | 1.322 |

Table 2: Mean squared errors between the ground truth and the reconstructions obtained by different methods.

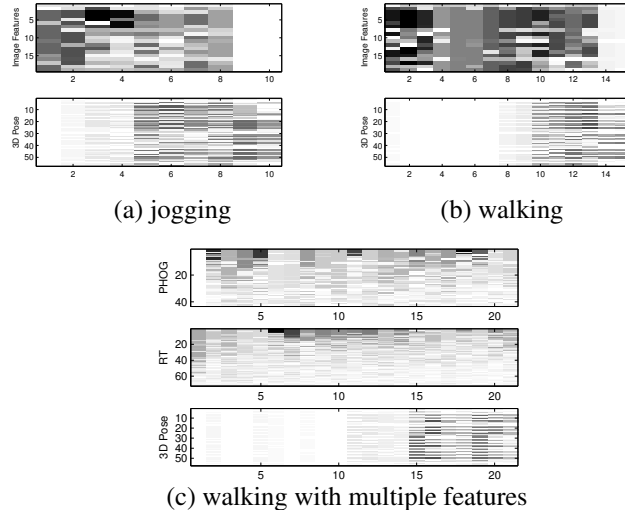

(a) jogging        (b) walking

(c) walking with multiple features

Figure 3: Dictionaries learned from the HumanEva data. Each column corresponds to a dictionary entry. (a) and (b) show the 2-view case, and (c) shows a three-view case. Note that in (c) our model found latent dimensions shared among all views, but also shared between the image features only.

models [21, 6, 15, 23], since they were shown to be outperformed by the FOLS-GPLVM [20] for human pose estimation.

To initialize the latent spaces for our model and for the FOLS-GPLVM, we proceeded similarly as for the toy example; We applied PCA on both views separately, as well as on the concatenated views, and retained the components representing 95% of the variance. In our case, we set $\alpha$ to be the concatenation of the corresponding PCA weights. For the FOLS-GPLVM, we initialized the shared latent space with the coefficients of the joint PCA, and the private spaces with those of the individual PCAs. We performed cross validation on the jogging data, and the optimal setting $\lambda = 0.01$ and $\gamma = 0.1$ was then fixed for all experiments.

At inference for human pose estimation, only one of the views (i.e., the images) is available. As shown in Section 2.4, our model provides a natural way to deal with this case by computing the latent variables from the image features first, and then recovering the 3D coordinates using the learned dictionary. For the FOLS-GPLVM, we followed the same strategy as in [20]; we computed the nearest-neighbor among the training examples in image feature space and took the corresponding shared and private latent variables that we mapped to the pose. No special care was required for the other baselines, since they explicitly take the images as inputs and the poses as outputs.

As a first case, we used hierarchical features [10] computed on the walking and jogging video sequences of the first subject seen from a single camera. As the subject moves in circles, we used the first loop to train our model, and the remaining ones for testing. Table 2 summarizes the mean squared reconstruction error for all the methods. Note that our approach yields a smaller error than the other methods. In Fig. 3(a,b), we show the factorization of the latent space obtained by our approach by displaying the learned dictionaries [2]. For the jogging case our algorithm automatically found a low-dimensional latent space of 10 dimensions, with a 4D private space for the image features, a 4D shared space, and a 2D private space for the 3D pose[3]. For the walking case, the

| Feature | Lin-Reg | GP-lin | GP-rbf | NN | FOLS | Our Method | $\lambda = 0$ | $\gamma = 0$ |
|---------|---------|--------|--------|------|-------|------------|---------------|--------------|
| PHOG | 1.190 | 1.167 | 0.839 | 1.279 | 1.277 | 0.778 | 2.886 | 0.863 |
| RT | 1.345 | 1.272 | 0.827 | 1.067 | 1.068 | 1.141 | 3.962 | 1.235 |
| PHOG+RT | 1.159 | 1.042 | 0.727 | 1.090 | 1.015 | 0.769 | 1.306 | 0.794 |

Table 3: Mean squared errors for different choices of image features. The last two columns show the result of our method while forcing one regularization term to be zero. See text for details.

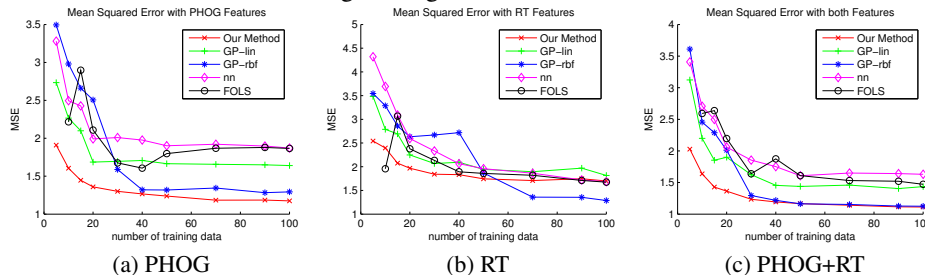

(a) PHOG  (b) RT  (c) PHOG+RT

Figure 4: Mean squared error as a function of the number of training examples using PHOG features only, RT features only, or both feature types simultaneously.

private space for the image features was found to be higher-dimensional. This can partially explain why the other methods did not perform as well as in the jogging case.

Next, we evaluated the performance of the same algorithms for different image features. In particular, we used randomized tree (RT) features generated by [19], and PHOG features [5]. For this case, we only considered the walking sequence and similarly trained the different methods using the first cycle and tested on the rest of the sequence. The top two rows of Table 3 show the results of the different approaches for the individual features. Note that, with the RT features that were designed to eliminate the ambiguities in pose estimation, GP regression with an RBF kernel performs slightly better than us. However, this result is outperformed by our model with PHOG features.

To show the ability of our method to model more than two views, we learned a latent space by simultaneously using RT features, PHOG features and 3D poses. The last row of Table 3 shows the corresponding reconstruction errors. In this case, we used the concatenated features as input to Lin-Reg, GP-lin and NN. For GP-rbf, we relied on kernel combination to predict the pose from multiple features. For the FOLS model, we applied the following inference strategy. We computed the NN in feature space for both features individually and took the mean of the corresponding shared latent variables. We then obtained the private part by computing the NN in shared space and taking the corresponding private variables. Note that this proved more accurate than using NN on a single view, or on the concatenated views. Also, notice in Table 3 that the performance drops when structured sparsity is only imposed on either $\mathbf{D}$'s or $\boldsymbol{\alpha}$, showing the advantage of our model over simple structured sparsity approaches. Fig. 3(c) depicts the dictionary found by our method. Note that our approach allowed us to find latent dimensions shared among all views, as well as shared among the image features only.

Finally, we studied the influence of the number of training examples on the performance of the different approaches. To this end, we varied the training set size from 5 to 100, and, for each size, randomly sampled 10 different training sets on the first walking cycle. In all cases, we kept the same test set as before. Fig. 4 shows the mean squared errors averaged over the 10 different sets as a function of the number of training examples. Note that, with small training sets, our method yields more accurate results than the baselines.

## 5    Conclusion

In this paper, we have proposed an approach to learning a latent space factorized into dimensions shared across subsets of the views and dimensions private to each individual view. To this end, we have proposed to exploit the notion of structured sparsity, and have shown that multi-view learning could be addressed by alternately solving two convex optimization problems. We have demonstrated the effectiveness of our approach on the task of estimating 3D human pose from image features. In the future, we intend to study the use of our model to other tasks, such as classification. To this end, we would extend our approach to incorporating an additional group sparsity regularizer on the latent variables to encode class membership.

## Footnotes

[1]In our paper, we define the $L_{p,q}$ norm of a matrix $\mathbf{A}$ to be the $p$-norm of the vector containing of the $q$-norms of the matrix rows, i.e., $\Vert\mathbf{A}\Vert_{p,q} = \left\Vert\,(\Vert\mathbf{A}_{1,\cdot}\Vert_q, \Vert\mathbf{A}_{2,\cdot}\Vert_q, \cdots, \Vert\mathbf{A}_{n,\cdot}\Vert_q)\,\right\Vert_p.$

[2]Note that the latent space per se is a dense, low-dimensional space, and whether a dimension is private or shared among multiple views is determined by the corresponding dictionary entries.

[3]A latent dimension is considered private if the norm of the corresponding dictionary entry in the other view is smaller than 10% of the average norm of the dictionary entries for that view.

# References

[1] C. Archambeau and F. Bach. Sparse probabilistic projections. In *Neural Information Processing Systems*, 2008.

[2] F. Bach, G. Lanckriet, and M. Jordan. Multiple kernel learning, conic duality, and the SMO algorithm. In *International Conference on Machine learning*. ACM New York, NY, USA, 2004.

[3] F. R. Bach and M. I. Jordan. A probabilistic interpretation of canonical correlation analysis. Technical Report 688, Department of Statistics, University of California, Berkeley, 2005.

[4] S. Bengio, F. Pereira, Y. Singer, and D. Strelow. Group sparse coding. *Neural Information Processing Systems*, 2009.

[5] A. Bosch, A. Zisserman, and X. Munoz. Image classification using random forests and ferns. In *International Conference on Computer Vision*, 2007.

[6] C. H. Ek, P. Torr, and N. Lawrence. Gaussian process latent variable models for human pose estimation. In *Joint Workshop on Machine Learning and Multimodal Interaction*, 2007.

[7] C. H. Ek, P. Torr, and N. Lawrence. Ambiguity modeling in latent spaces. In *Joint Workshop on Machine Learning and Multimodal Interaction*, 2008.

[8] A. Geiger, R. Urtasun, and T. Darrell. Rank priors for continuous non-linear dimensionality reduction. In *Conference on Computer Vision and Pattern Recognition*, 2009.

[9] R. Jenatton, G. Obozinski, and F. Bach. Structured sparse principal component analysis. In *International Conference on Artificial Intelligence and Statistics*, Sardinia, Italy, May 2010.

[10] A. Kanaujia, C. Sminchisescu, and D. N. Metaxas. Semi-supervised hierarchical models for 3d human pose reconstruction. In *Conference on Computer Vision and Pattern Recognition*, 2007.

[11] A. Klami and S. Kaski. Probabilistic approach to detecting dependencies between data sets. *Neurocomputing*, 72:39–46, 2008.

[12] M. Kuss and T. Graepel. The geometry of kernel canonical correlation analysis. Technical Report TR-108, Max Planck Institute for Biological Cybernetics, Tübingen, Germany, 2003.

[13] H. Lee, A. Battle, R. Raina, and A. Y. Ng. Efficient sparse coding algorithms. In *Neural Information Processing Systems*, 2006.

[14] G. Leen. *Context assisted information extraction*. PhD thesis, University the of West of Scotland, University of the West of Scotland, High Street, Paisley PA1 2BE, Scotland, 2008.

[15] R. Navaratnam, A. Fitzgibbon, and R. Cipolla. The Joint Manifold Model for Semi-supervised Multivalued Regression. In *International Conference on Computer Vision*, Rio, Brazil, October 2007.

[16] B. Olshausen and D. Field. Emergence of simple-cell receptive field properties by learning a sparse code for natural images. *Nature*, 381:607–609, 1996.

[17] A. Quattoni, X. Carreras, M. Collins, and T. Darrell. An efficient projection for l1,infinity regularization. In *International Conference on Machine Learning*, 2009.

[18] A. Quattoni, M. Collins, and T. Darrell. Transfer learning for image classification with sparse prototype representations. In *Conference on Computer Vision and Pattern Recognition*, 2008.

[19] G. Rogez, J. Rihan, S. Ramalingam, C. Orrite, and P. Torr. Randomized Trees for Human Pose Detection. In *Conference on Computer Vision and Pattern Recognition*, 2008.

[20] M. Salzmann, C.-H. Ek, R. Urtasun, and T. Darrell. Factorized orthogonal latent spaces. In *International Conference on Artificial Intelligence and Statistics*, Sardinia, Italy, May 2010.

[21] A. P. Shon, K. Grochow, A. Hertzmann, and R. P. N. Rao. Learning shared latent structure for image synthesis and robotic imitation. In *Neural Information Processing Systems*, pages 1233–1240, 2006.

[22] L. Sigal and M. J. Black. Humaneva: Synchronized video and motion capture dataset for evaluation of articulated human motion. Technical Report CS-06-08, Brown University, 2006.

[23] L. Sigal, R. Memisevic, and D. J. Fleet. Shared kernel information embedding for discriminative inference. In *Conference on Computer Vision and Pattern Recognition*, 2009.

[24] S. Sonnenburg, G. Rätsch, C. Schäfer, and B. Schölkopf. Large scale multiple kernel learning. *The Journal of Machine Learning Research*, 7:1531–1565, 2006.

[25] R. Tibshirani. Regression shrinkage and selection via the lasso. *Journal of the Royal Statistical Society, Series B*, 58:267–288, 1996.

[26] M. Yuan and Y. Lin. Model selection and estimation in regression with grouped variables. *Journal of the Royal Statistical Society, Series B*, 68:49–67, 2006.

